# Dependent Gaussian Processes

**Phillip Boyle and Marcus Frean**
School of Mathematical and Computing Sciences
Victoria University of Wellington,
Wellington, New Zealand
{pkboyle,marcus}@mcs.vuw.ac.nz

## Abstract

Gaussian processes are usually parameterised in terms of their covariance functions. However, this makes it difficult to deal with multiple outputs, because ensuring that the covariance matrix is positive definite is problematic. An alternative formulation is to treat Gaussian processes as white noise sources convolved with smoothing kernels, and to parameterise the kernel instead. Using this, we extend Gaussian processes to handle multiple, coupled outputs.

## 1 Introduction

Gaussian process regression has many desirable properties, such as ease of obtaining and expressing uncertainty in predictions, the ability to capture a wide variety of behaviour through a simple parameterisation, and a natural Bayesian interpretation [15, 4, 9]. Because of this they have been suggested as replacements for supervised neural networks in non-linear regression [8, 18], extended to handle classification tasks [11, 17, 6], and used in a variety of other ways (e.g. [16, 14]). A Gaussian process (GP), as a set of jointly Gaussian random variables, is completely characterised by a covariance matrix with entries determined by a covariance function. Traditionally, such models have been specified by parameterising the covariance function (*i.e.* a function specifying the covariance of output values given any two input vectors). In general this needs to be a positive definite function to ensure positive definiteness of the covariance matrix.

Most GP implementations model only a single output variable. Attempts to handle multiple outputs generally involve using an independent model for each output - a method known as multi-kriging [18] - but such models cannot capture the structure in outputs that covary. As an example, consider the two tightly coupled outputs shown at the top of Figure 2, in which one output is simply a shifted version of the other. Here we have detailed knowledge of output 1, but sampling of output 2 is sparse. A model that treats the outputs as independent cannot exploit their obvious similarity - intuitively, we should make predictions about output 2 using what we learn from both output 1 *and* 2.

Joint predictions are possible (e.g. co-kriging [3]) but are problematic in that it is not clear how covariance functions should be defined [5]. Although there are many known positive definite autocovariance functions (e.g. Gaussians and many others [1, 9]), it is difficult to define cross-covariance functions that result in positive definite covariance matrices. Contrast this to neural network modelling, where the handling of multiple outputs is routine.

An alternative to directly parameterising covariance functions is to treat GPs as the outputs of stable linear filters. For a linear filter, the output in response to an input $x(t)$ is $y(t) = h(t) \star x(t) = \int_{-\infty}^{\infty} h(t - \tau)x(\tau)d\tau$, where $h(t)$ defines the impulse response of the filter and $\star$ denotes convolution. Provided the linear filter is stable and $x(t)$ is Gaussian white noise, the output process $y(t)$ is necessarily a Gaussian process. It is also possible to characterise $p$-dimensional stable linear filters, with $M$-inputs and $N$-outputs, by a set of $M \times N$ impulse responses. In general, the resulting $N$ outputs are dependent Gaussian processes. Now we can model multiple dependent outputs by parameterising the set of impulse responses for a multiple output linear filter, and inferring the parameter values from data that we observe. Instead of specifying and parameterising positive definite covariance functions, we now specify and parameterise impulse responses. The only restriction is that the filter be linear and stable, and this is achieved by requiring the impulse responses to be absolutely integrable.

Constructing GPs by stimulating linear filters with Gaussian noise is equivalent to constructing GPs through kernel convolutions. A Gaussian process $V(s)$ can be constructed over a region $\mathcal{S}$ by convolving a continuous white noise process $X(s)$ with a smoothing kernel $h(s)$, $V(s) = h(s) \star X(s)$ for $s \in \mathcal{S}$, [7]. To this can be added a second white noise source, representing measurement uncertainty, and together this gives a model for observations $Y$. This view of GPs is shown in graphical form in Figure 1($a$). The convolution approach has been used to formulate flexible nonstationary covariance functions [13, 12]. Furthermore, this idea can be extended to model multiple dependent output processes by assuming a single common latent process [7]. For example, two dependent processes $V_1(s)$ and $V_2(s)$ are constructed from a shared dependence on $X(s)$ for $s \in \mathcal{S}_0$, as follows

$$V_1(s) = \int_{\mathcal{S}_0 \cup \mathcal{S}_1} h_1(s - \lambda)X(\lambda)d\lambda \quad \text{and} \quad V_2(s) = \int_{\mathcal{S}_0 \cup \mathcal{S}_2} h_2(s - \lambda)X(\lambda)d\lambda$$

where $\mathcal{S} = \mathcal{S}_0 \cup \mathcal{S}_1 \cup \mathcal{S}_2$ is a union of disjoint subspaces. $V_1(s)$ is dependent on $X(s), s \in \mathcal{S}_1$ but not $X(s), s \in \mathcal{S}_2$. Similarly, $V_2(s)$ is dependent on $X(s), s \in \mathcal{S}_2$ but not $X(s), s \in \mathcal{S}_1$. This allows $V_1(s)$ and $V_2(s)$ to possess independent components.

In this paper, we model multiple outputs somewhat differently to [7]. Instead of assuming a single latent process defined over a union of subspaces, we assume multiple latent processes, each defined over $\Re^p$. Some outputs may be dependent through a shared reliance on common latent processes, and some outputs may possess unique, independent features through a connection to a latent process that affects no other output.

## 2 Two Dependent Outputs

Consider two outputs $Y_1(s)$ and $Y_2(s)$ over a region $\Re^p$, where $s \in \Re^p$. We have $N_1$ observations of output 1 and $N_2$ observations of output 2, giving us data $\mathcal{D}_1 = \{s_{1,i}, y_{1,i}\}_{i=1}^{N_1}$ and $\mathcal{D}_2 = \{s_{2,i}, y_{2,i}\}_{i=1}^{N_2}$. We wish to learn a model from the combined data $\mathcal{D} = \{\mathcal{D}_1, \mathcal{D}_2\}$ in order to predict $Y_1(s')$ or $Y_2(s')$, for $s' \in \Re^p$. As shown in Figure 1($b$), we can model each output as the linear sum of three stationary Gaussian processes. One of these ($V$) arises from a noise source unique to that output, under convolution with a kernel $h$. A second ($U$) is similar, but arises from a separate noise source $X_0$ that influences *both* outputs (although via different kernels, $k$). The third is additive noise as before.

Thus we have $Y_i(s) = U_i(s) + V_i(s) + W_i(s)$, where $W_i(s)$ is a stationary Gaussian white noise process with variance, $\sigma_i^2$, $X_0(s), X_1(s)$ and $X_2(s)$ are independent stationary Gaussian white noise processes, $U_1(s), U_2(s), V_1(s)$ and $V_2(s)$ are Gaussian processes given by $U_i(s) = k_i(s) \star X_0(s)$ and $V_i(s) = h_i(s) \star X_i(s)$.

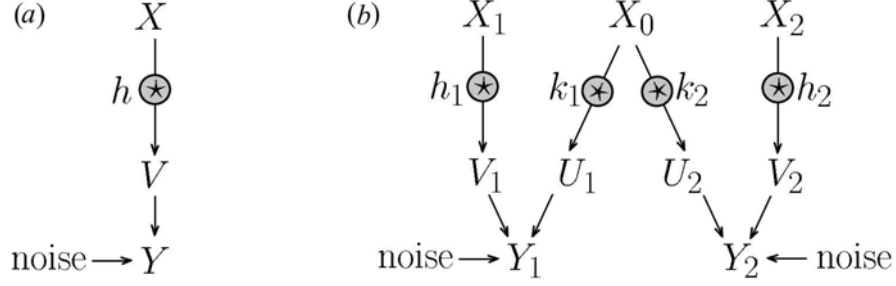

Figure 1: (*a*) Gaussian process prior for a single output. The output $Y$ is the sum of two Gaussian white noise processes, one of which has been convolved ($\star$) with a kernel ($h$). (*b*) The model for two dependent outputs $Y_1$ and $Y_2$. All of $X_0, X_1, X_2$ and the "noise" contributions are independent Gaussian white noise sources. Notice that if $X_0$ is forced to zero $Y_1$ and $Y_2$ become independent processes as in (*a*) - we use this as a control model.

The $k_1, k_2, h_1, h_2$ are parameterised Gaussian kernels where $k_1(s) = v_1 \exp\left(-\frac{1}{2}s^T A_1 s\right)$, $k_2(s) = v_2 \exp\left(-\frac{1}{2}(s-\mu)^T A_2(s-\mu)\right)$, and $h_i(s) = w_i \exp\left(-\frac{1}{2}s^T B_i s\right)$. Note that $k_2(s)$ is offset from zero by $\mu$ to allow modelling of outputs that are coupled and translated relative to one another.

We wish to derive the set of functions $C_{ij}^Y(d)$ that define the autocovariance ($i = j$) and cross-covariance ($i \neq j$) between the outputs $i$ and $j$, for a given separation $d$ between arbitrary inputs $s_a$ and $s_b$. By solving a convolution integral, $C_{ij}^Y(d)$ can be expressed in a closed form [2], and is fully determined by the parameters of the Gaussian kernels and the noise variances $\sigma_1^2$ and $\sigma_2^2$ as follows:

$$C_{11}^Y(d) = C_{11}^U(d) + C_{11}^V(d) + \delta_{ab}\sigma_1^2 \qquad\qquad C_{12}^Y(d) = C_{12}^U(d)$$
$$C_{22}^Y(d) = C_{22}^U(d) + C_{22}^V(d) + \delta_{ab}\sigma_2^2 \qquad\qquad C_{21}^Y(d) = C_{21}^U(d)$$

where

$$C_{ii}^U(d) = \frac{\pi^{\frac{p}{2}} v_i^2}{\sqrt{|A_i|}} \exp\left(-\frac{1}{4} d^T A_i d\right)$$

$$C_{12}^U(d) = \frac{(2\pi)^{\frac{p}{2}} v_1 v_2}{\sqrt{|A_1 + A_2|}} \exp\left(-\frac{1}{2}(d-\mu)^T \Sigma(d-\mu)\right)$$

$$C_{21}^U(d) = \frac{(2\pi)^{\frac{p}{2}} v_1 v_2}{\sqrt{|A_1 + A_2|}} \exp\left(-\frac{1}{2}(d+\mu)^T \Sigma(d+\mu)\right) = C_{12}^U(-d)$$

$$C_{ii}^V(d) = \frac{\pi^{\frac{p}{2}} w_i^2}{\sqrt{|B_i|}} \exp\left(-\frac{1}{4} d^T B_i d\right)$$

with $\Sigma = A_1(A_1 + A_2)^{-1}A_2 = A_2(A_1 + A_2)^{-1}A_1$.

Given $C_{ij}^Y(d)$ then, we can construct the covariance matrices $\mathbf{C}_{11}, \mathbf{C}_{12}, \mathbf{C}_{21}$, and $\mathbf{C}_{22}$ as follows

$$\mathbf{C}_{ij} = \begin{bmatrix} C_{ij}^Y(s_{i,1} - s_{j,1}) & \cdots & C_{ij}^Y(s_{i,1} - s_{j,N_j}) \\ \vdots & \ddots & \vdots \\ C_{ij}^Y(s_{i,N_i} - s_{j,1}) & \cdots & C_{ij}^Y(s_{i,N_i} - s_{j,N_j}) \end{bmatrix} \tag{1}$$

Together these define the positive definite symmetric covariance matrix $\mathbf{C}$ for the *combined* output data $\mathcal{D}$:

$$\mathbf{C} = \begin{bmatrix} \mathbf{C}_{11} & \mathbf{C}_{12} \\ \mathbf{C}_{21} & \mathbf{C}_{22} \end{bmatrix} \tag{2}$$

We define a set of hyperparameters $\boldsymbol{\Theta}$ that parameterise $\{v_1, v_2, w_1, w_2, A_1, A_2, B_1, B_2, \mu, \sigma_1, \sigma_2\}$. Now, we can calculate the likelihood

$$\mathcal{L} = -\frac{1}{2}\log|\mathbf{C}| - \frac{1}{2}\mathbf{y}^T\mathbf{C}^{-1}\mathbf{y} - \frac{N_1 + N_2}{2}\log 2\pi$$

$$\text{where} \quad \mathbf{y}^T = [y_{1,1} \quad \cdots \quad y_{1,N_1} \quad y_{2,1} \quad \cdots \quad y_{2,N_2}]$$

and $\mathbf{C}$ is a function of $\boldsymbol{\Theta}$ and $\mathcal{D}$.

Learning a model now corresponds to either maximising the likelihood $\mathcal{L}$, or maximising the posterior probability $P(\boldsymbol{\Theta}\,|\,\mathcal{D})$. Alternatively, we can simulate the predictive distribution for $y$ by taking samples from the joint $P(\mathbf{y}, \boldsymbol{\Theta}\,|\,\mathcal{D})$, using Markov Chain Monte Carlo methods [10].

The predictive distribution at a point $s'$ on output $i$ given $\boldsymbol{\Theta}$ and $\mathcal{D}$ is Gaussian with mean $\hat{y}'$ and variance $\sigma_{\hat{y}'}^2$ given by

$$\hat{y}' = \mathbf{k}^T\mathbf{C}^{-1}\mathbf{y}$$

$$\text{and} \quad \sigma_{\hat{y}'}^2 = \kappa - \mathbf{k}^T\mathbf{C}^{-1}\mathbf{k}$$

$$\text{where} \quad \kappa = \mathbf{C}_{ii}^Y(0) = v_i^2 + w_i^2 + \sigma_i^2$$

$$\text{and} \quad \mathbf{k} = \left[C_{i1}^Y(s' - s_{1,1})\ldots C_{i1}^Y(s' - s_{1,N_1}) \quad C_{i2}^Y(s' - s_{2,1})\ldots C_{i2}^Y(s' - s_{2,N_2})\right]^T$$

## 2.1 Example 1 - Strongly dependent outputs over 1d input space

Consider two outputs, observed over a 1d input space. Let $A_i = \exp(f_i), \quad B_i = \exp(g_i)$, and $\sigma_i = \exp(\beta_i)$. Our hyperparameters are $\boldsymbol{\Theta} = \{v_1, v_2, w_1, w_2, f_1, f_2, g_1, g_2, \mu, \beta_1, \beta_2\}$ where each element of $\boldsymbol{\Theta}$ is a scalar. As in [2] we set Gaussian priors over $\boldsymbol{\Theta}$.

We generated $N = 48$ data points by taking $N_1 = 32$ samples from output 1 and $N_2 = 16$ samples from output 2. The samples from output 1 were linearly spaced in the interval $[-1, 1]$ and those from output 2 were uniformly spaced in the region $[-1, -0.15] \cup [0.65, 1]$. All samples were taken under additive Gaussian noise, $\sigma = 0.025$. To build our model, we maximised $P(\boldsymbol{\Theta}|\mathcal{D}) \propto P(\mathcal{D}\,|\,\boldsymbol{\Theta})\,P(\boldsymbol{\Theta})$ using a multistart conjugate gradient algorithm, with 5 starts, sampling from $P(\boldsymbol{\Theta})$ for initial conditions.

The resulting dependent model is shown in Figure 2 along with an independent (control) model with no coupling (see Figure 1). Observe that the dependent model has learned the coupling and translation between the outputs, and has filled in output 2 where samples are missing. The control model cannot achieve such infilling as it is consists of two independent Gaussian processes.

## 2.2 Example 2 - Strongly dependent outputs over 2d input space

Consider two outputs, observed over a 2d input space. Let

$$A_i = \frac{1}{\alpha_i^2}\mathbf{I} \qquad B_i = \frac{1}{\tau_i^2}\mathbf{I} \qquad \text{where } \mathbf{I} \text{ is the identity matrix.}$$

Furthermore, let $\sigma_i = \exp(\beta_i)$. In this toy example, we set $\mu = 0$, so our hyperparameters become $\boldsymbol{\Theta} = \{v_1, v_2, w_1, w_2, \alpha_1, \alpha_2, \tau_1, \tau_2 \beta_1, \beta_2\}$ where each element of $\boldsymbol{\Theta}$ is a scalar. Again, we set Gaussian priors over $\boldsymbol{\Theta}$.

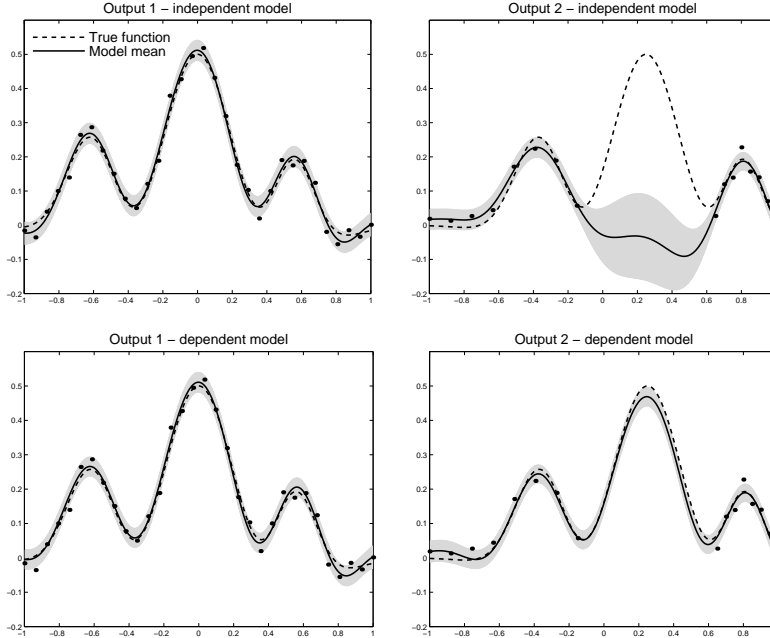

Figure 2: Strongly dependent outputs where output 2 is simply a translated version of output 1, with independent Gaussian noise, $\sigma = 0.025$. The solid lines represent the model, the dotted lines are the true function, and the dots are samples. The shaded regions represent $1\sigma$ error bars for the model prediction. *(top)* Independent model of the two outputs. *(bottom)* Dependent model.

We generated 117 data points by taking 81 samples from output 1 and 36 samples from output 2. Both sets of samples formed uniform lattices over the region $[-0.9, 0.9] \otimes [-0.9, 0.9]$ and were taken with additive Gaussian noise, $\sigma = 0.025$. To build our model, we maximised $P(\Theta|\mathcal{D})$ as before.

The dependent model is shown in Figure 3 along with an independent control model. The dependent model has filled in output 2 where samples are missing. Again, the control model cannot achieve such in-filling as it is consists of two independent Gaussian processes.

## 3  Time Series Forecasting

Consider the observation of multiple time series, where some of the series lead or predict the others. We simulated a set of three time series for 100 steps each (figure 4) where series 3 was positively coupled to a lagged version of series 1 ($lag = 0.5$) and negatively coupled to a lagged version of series 2 ($lag = 0.6$). Given the 300 observations, we built a dependent GP model of the three time series and compared them with independent GP models. The dependent GP model incorporated a prior belief that series 3 was coupled to series 1 and 2, with the lags unknown. The independent GP model assumed no coupling between its outputs, and consisted of three independent GP models. We queried the models for forecasts of the future 10 values of series 3. It is clear from figure 4 that the dependent GP model does a far better job at forecasting the dependent series 3. The independent model becomes inaccurate after just a few time steps into the future. This inaccuracy is expected as knowledge of series 1 and 2 is required to accurately predict series 3. The

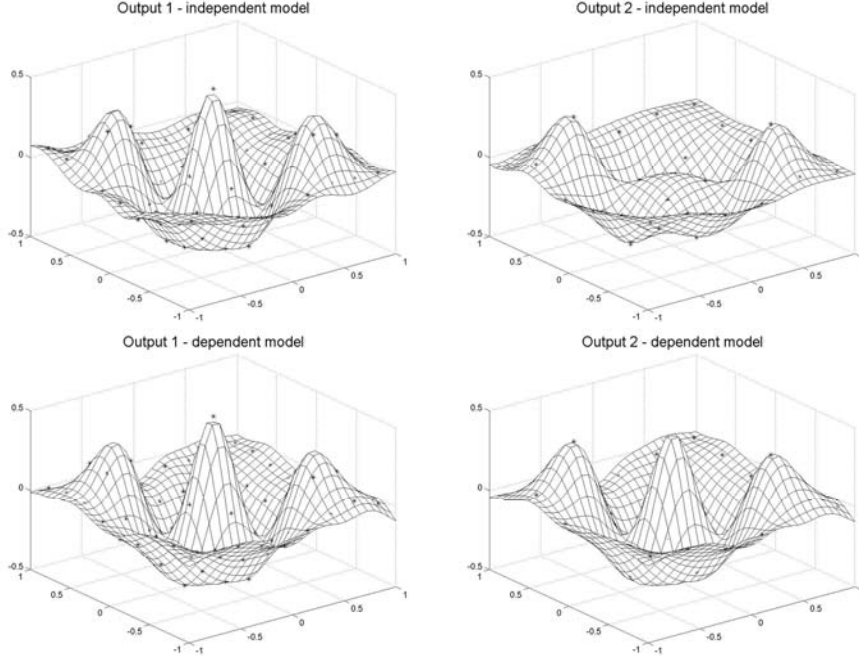

Figure 3: Strongly dependent outputs where output 2 is simply a copy of output 1, with independent Gaussian noise. *(top)* Independent model of the two outputs. *(bottom)* Dependent model. Output 1 is modelled well by both models. Output 2 is modelled well only by the dependent model

dependent GP model performs well as it has learned that series 3 is positively coupled to a lagged version of series 1 and negatively coupled to a lagged version of series 2.

## 4 Multiple Outputs and Non-stationary Kernels

The convolution framework described here for constructing GPs can be extended to build models capable of modelling $N$-outputs, each defined over a $p$-dimensional input space. In general, we can define a model where we assume $M$-independent Gaussian white noise processes $X_1(s) \ldots X_M(s)$, $N$-outputs $U_1(s) \ldots U_N(s)$, and $M \times N$ kernels $\{\{k_{mn}(s)\}_{m=1}^{M}\}_{n=1}^{N}$ where $s \in \Re^p$. The autocovariance $(i = j)$ and cross-covariance $(i \neq j)$ functions between output processes $i$ and $j$ become

$$C_{ij}^{U}(d) = \sum_{m=1}^{M} \int_{\Re^p} k_{mi}(s) k_{mj}(s+d) ds \tag{3}$$

and the matrix defined by equation 2 is extended in the obvious way.

The kernels used in (3) need not be Gaussian, and need not be spatially invariant, or stationary. We require kernels that are absolutely integrable, $\int_{-\infty}^{\infty} \ldots \int_{-\infty}^{\infty} |k(s)| d^p s < \infty$. This provides a large degree of flexibility, and is an easy condition to uphold. It would seem that an absolutely integrable kernel would be easier to define and parameterise than a positive definite function. On the other hand, we require a closed form of $C_{ij}^{Y}(d)$ and this may not be attainable for some non-Gaussian kernels.

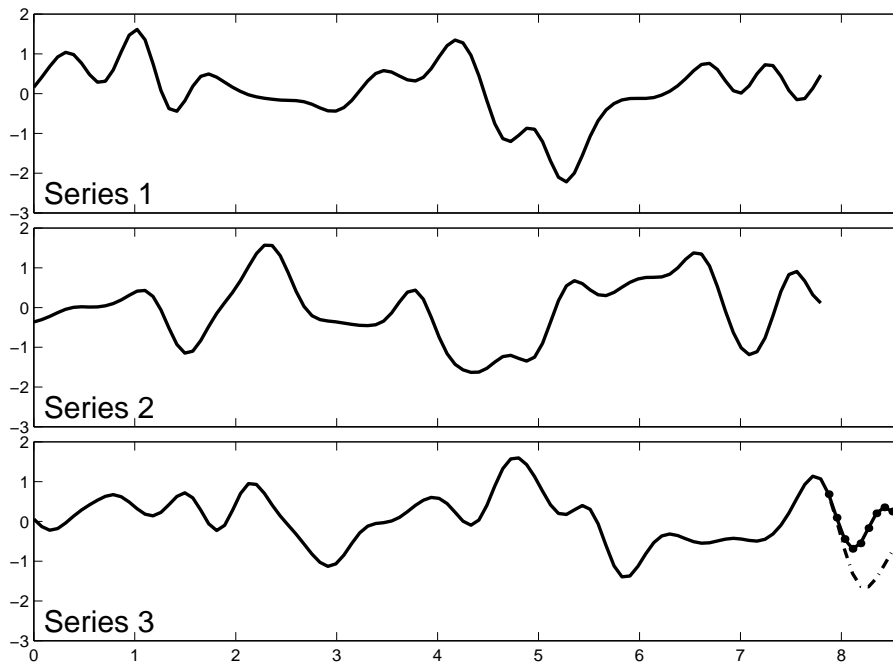

Figure 4: Three coupled time series, where series 1 and series 2 predict series 3. Forecasting for series 3 begins after 100 time steps where $t = 7.8$. The dependent model forecast is shown with a solid line, and the independent (control) forecast is shown with a broken line. The dependent model does a far better job at forecasting the next 10 steps of series 3 (black dots).

## 5  Conclusion

We have shown how the Gaussian Process framework can be extended to multiple output variables without assuming them to be independent. Multiple processes can be handled by inferring convolution kernels instead of covariance functions. This makes it easy to construct the required positive definite covariance matrices for covarying outputs.

One application of this work is to learn the spatial translations between outputs. However the framework developed here is more general than this, as it can model data that arises from multiple sources, only some of which are shared. Our examples show the infilling of sparsely sampled regions that becomes possible in a model that permits coupling between outputs. Another application is the forecasting of dependent time series. Our example shows how learning couplings between multiple time series may aid in forecasting, particularly when the series to be forecast is dependent on previous or current values of other series.

Dependent Gaussian processes should be particularly valuable in cases where one output is expensive to sample, but covaries strongly with a second that is cheap. By inferring both the coupling and the independent aspects of the data, the cheap observations can be used as a proxy for the expensive ones.

# References

[1] ABRAHAMSEN, P. A review of gaussian random fields and correlation functions. Tech. Rep. 917, Norwegian Computing Center, Box 114, Blindern, N-0314 Oslo, Norway, 1997.

[2] BOYLE, P., AND FREAN, M. Multiple-output gaussian process regression. Tech. rep., Victoria University of Wellington, 2004.

[3] CRESSIE, N. *Statistics for Spatial Data*. Wiley, 1993.

[4] GIBBS, M. *Bayesian Gaussian Processes for Classification and Regression*. PhD thesis, University of Cambridge, Cambridge, U.K., 1997.

[5] GIBBS, M., AND MACKAY, D. J. Efficient implementation of gaussian processes. www.inference.phy.cam.ac.uk/mackay/abstracts/gpros.html, 1996.

[6] GIBBS, M. N., AND MACKAY, D. J. Variational gaussian process classifiers. *IEEE Trans. on Neural Networks 11*, 6 (2000), 1458–1464.

[7] HIGDON, D. Space and space-time modelling using process convolutions. In *Quantitative methods for current environmental issues* (2002), C. Anderson, V. Barnett, P. Chatwin, and A. El-Shaarawi, Eds., Springer Verlag, pp. 37–56.

[8] MACKAY, D. J. Gaussian processes: A replacement for supervised neural networks? *In NIPS97 Tutorial*, 1997.

[9] MACKAY, D. J. *Information theory, inference, and learning algorithms*. Cambridge University Press, 2003.

[10] NEAL, R. Probabilistic inference using markov chain monte carlo methods. Tech. Report CRG-TR-93-1, Dept. of Computer Science, Univ. of Toronto, 1993.

[11] NEAL, R. Monte carlo implementation of gaussian process models for bayesian regression and classification. Tech. Rep. CRG-TR-97-2, Dept. of Computer Science, Univ. of Toronto, 1997.

[12] PACIOREK, C. *Nonstationary Gaussian processes for regression and spatial modelling*. PhD thesis, Carnegie Mellon University, Pittsburgh, Pennsylvania, U.S.A., 2003.

[13] PACIOREK, C., AND SCHERVISH, M. Nonstationary covariance functions for gaussian process regression. *Submitted to NIPS*, 2004.

[14] RASMUSSEN, C., AND KUSS, M. Gaussian processes in reinforcement learning. In *Advances in Neural Information Processing Systems* (2004), vol. 16.

[15] RASMUSSEN, C. E. *Evaluation of Gaussian Processes and other methods for Non-Linear Regression*. PhD thesis, Graduate Department of Computer Science, University of Toronto, 1996.

[16] TIPPING, M. E., AND BISHOP, C. M. Bayesian image super-resolution. In *Advances in Neural Information Processing Systems* (2002), S. Becker S., Thrun and K. Obermayer, Eds., vol. 15, pp. 1303 – 1310.

[17] WILLIAMS, C. K., AND BARBER, D. Bayesian classification with gaussian processes. *IEEE trans. Pattern Analysis and Machine Intelligence 20*, 12 (1998), 1342 – 1351.

[18] WILLIAMS, C. K., AND RASMUSSEN, C. E. Gaussian processes for regression. In *Advances in Neural Information Processing Systems* (1996), D. Touretzsky, M. Mozer, and M. Hasselmo, Eds., vol. 8.
